# LEARNING UNAMBIGUOUS REDUCED SEQUENCE DESCRIPTIONS

**Jürgen Schmidhuber**
Dept. of Computer Science
University of Colorado
Campus Box 430
Boulder, CO 80309, USA
yirgan@cs.colorado.edu

## Abstract

Do you want your neural net algorithm to learn sequences? Do not limit yourself to conventional gradient descent (or approximations thereof). Instead, use your sequence learning algorithm (any will do) to implement the following method for history compression. No matter what your final goals are, train a network to predict its next input from the previous ones. Since only unpredictable inputs convey new information, ignore all predictable inputs but let all unexpected inputs (plus information about the time step at which they occurred) become inputs to a higher-level network of the same kind (working on a slower, self-adjusting time scale). Go on building a hierarchy of such networks. This principle reduces the descriptions of event sequences *without loss of information*, thus easing supervised or reinforcement learning tasks. Alternatively, you may use two recurrent networks to collapse a multi-level predictor hierarchy into a single recurrent net. Experiments show that systems based on these principles can require less computation per time step *and* many fewer training sequences than conventional training algorithms for recurrent nets. Finally you can modify the above method such that predictability is not defined in a yes-or-no fashion but in a continuous fashion.

# 1   INTRODUCTION

The following methods for supervised sequence learning have been proposed: Simple recurrent nets [7][3], time-delay nets (e.g. [2]), sequential recursive auto-associative memories [16], back-propagation through time or BPTT [21] [30] [33], Mozer's 'focused back-prop' algorithm [10], the IID- or RTRL-algorithm [19][1][34], its accelerated versions [32][35][25], the recent fast-weight algorithm [27], higher-order networks [5], as well as continuous time methods equivalent to some of the above [14][15][4]. The following methods for sequence learning by reinforcement learning have been proposed: Extended REINFORCE algorithms [31], the neural bucket brigade algorithm [22], recurrent networks adjusted by adaptive critics [23](see also [8]), buffer-based systems [13], and networks of hierarchically organized neuron-like "bions" [18].

With the exception of [18] and [13], these approaches waste resources and limit efficiency by focusing on *every* input instead of focusing only on *relevant* inputs. Many of these methods have a second drawback as well: The longer the time lag between an event and the occurrence of a related error the less information is carried by the corresponding error information wandering 'back into time' (see [6] for a more detailed analysis). [11], [12] and [20] have addressed the latter problem but not the former. The system described by [18] on the other hand addresses both problems, but in a manner much different from that presented here.

# 2   HISTORY COMPRESSION

A major contribution of this work is an adaptive method for removing redundant information from sequences. This principle can be implemented with the help of any of the methods mentioned in the introduction.

Consider a deterministic discrete time predictor (not necessarily a neural network) whose state at time $t$ of sequence $p$ is described by an environmental input vector $x^p(t)$, an internal state vector $h^p(t)$, and an output vector $z^p(t)$. The environment may be non-deterministic. At time 0, the predictor starts with $x^p(0)$ and an internal start state $h^p(0)$. At time $t \geq 0$, the predictor computes

$$z^p(t) = f(x^p(t), h^p(t)).$$

At time $t > 0$, the predictor furthermore computes

$$h^p(t) = g(x^p(t-1), h^p(t-1)).$$

*All* information about the input at a given time $t_x$ can be reconstructed from $t_x, f, g, x^p(0), h^p(0)$, and the pairs $(t_s, x^p(t_s))$ for which $0 < t_s \leq t_x$ and $z^p(t_s - 1) \neq x^p(t_s)$. This is because if $z^p(t) = x^p(t+1)$ at a given time $t$, then the predictor is able to predict the next input from the previous ones. The new input is *derivable* by means of $f$ and $g$.

Information about the observed input sequence can be even further compressed beyond just the unpredicted input vectors $x^p(t_s)$. It suffices to know only those *elements* of the vectors $x^p(t_s)$ that were not correctly predicted.

This observation implies that we can discriminate one sequence from another by knowing *just the unpredicted inputs and the corresponding time steps at which they*

*occurred.* No information is lost if we ignore the expected inputs. We do not even have to know $f$ and $g$. I call this *the principle of history compression.*

From a theoretical point of view it is important to know at what time an unexpected input occurs; otherwise there will be a potential for ambiguities: Two different input sequences may lead to the same shorter sequence of unpredicted inputs. With many practical tasks, however, there is no need for knowing the critical time steps (see section 5).

# 3  SELF-ORGANIZING PREDICTOR HIERARCHY

Using the principle of history compression we can build a self-organizing hierarchical neural 'chunking' system[1]. The basic task can be formulated as a prediction task. At a given time step the goal is to predict the next input from previous inputs. If there are external target vectors at certain time steps then they are simply treated as another part of the input to be predicted.

The architecture is a hierarchy of predictors, the input to each level of the hierarchy is coming from the previous level. $P_i$ denotes the $i$th level network which is trained to *predict its own next input from its previous inputs*[2]. We take $P_i$ to be one of the conventional dynamic recurrent neural networks mentioned in the introduction; however, it might be some other adaptive sequence processing device as well[3].

At each time step the input of the lowest-level recurrent predictor $P_0$ is the current external input. We create a new higher-level adaptive predictor $P_{s+1}$ whenever the adaptive predictor at the previous level, $P_s$, stops improving its predictions. When this happens the weight-changing mechanism of $P_s$ is switched off (to exclude potential instabilities caused by ongoing modifications of the lower-level predictors). If at a given time step $P_s$ ($s \geq 0$) fails to predict its next input (or if we are at the beginning of a training sequence which usually is not predictable either) then $P_{s+1}$ will receive as input the concatenation of this next input of $P_s$ *plus a unique representation of the corresponding time step*[4]; the activations of $P_{s+1}$'s hidden and output units will be updated. Otherwise $P_{s+1}$ will not perform an activation update. This procedure ensures that $P_{s+1}$ is fed with an *unambiguous* reduced description[5] of the input sequence observed by $P_s$. This is theoretically justified by the principle of history compression.

In general, $P_{s+1}$ will receive fewer inputs over time than $P_s$. With existing learning

algorithms, the higher-level predictor should have less difficulties in learning to predict the critical inputs than the lower-level predictor. This is because $P_{s+1}$'s 'credit assignment paths' will often be short compared to those of $P_s$. This will happen if the incoming inputs carry global temporal structure which has not yet been discovered by $P_s$. (See also [18] for a related approach to the problem of credit assignment in reinforcement learning.)

This method is a simplification and an improvement of the recent chunking method described by [24].

A multi-level predictor hierarchy is a rather safe way of learning to deal with sequences with multi-level temporal structure (e.g speech). Experiments have shown that multi-level predictors can quickly learn tasks which are practically unlearnable by conventional recurrent networks, e.g. [6].

## 4    COLLAPSING THE HIERARCHY

One disadvantage of a predictor hierarchy as above is that it is not known in advance how many levels will be needed. Another disadvantage is that levels are explicitly separated from each other. It may be possible, however, to collapse the hierarchy into a single network as outlined in this section. See details in [26].

We need two conventional recurrent networks: The *automatizer* A and the *chunker* C, which correspond to a distinction between automatic and attended events. (See also [13] and [17] which describe a similar distinction in the context of reinforcement learning). At each time step A receives the current external input. A's error function is threefold: One term forces it to emit certain desired target outputs at certain times. If there is a target, then it becomes part of the next input. The second term forces A at every time step to predict its own next non-target input. The third (crucial) term will be explained below.

If and only if A makes an error concerning the first and second term of its error function, the unpredicted input (including a potentially available teaching vector) *along with a unique representation of the current time step* will become the new input to C. Before this new input can be processed, C (whose last input may have occurred many time steps earlier) is trained to predict this higher-level input from its current internal state and its last input (employing a conventional recurrent net algorithm). After this, C performs an activation update which contributes to a higher level internal representation of the input history. Note that according to the principle of history compression C is fed with an *unambiguous reduced description of the input history*. The information deducible by means of A's predictions can be considered as *redundant*. (The beginning of an episode usually is not predictable, therefore it has to be fed to the chunking level, too.)

Since C's 'credit assignment paths' will often be short compared to those of A, C will often be able to develop useful internal representations of previous unexpected input events. Due to the final term of its error function, A will be forced to reproduce these internal representations, *by predicting C's state*. Therefore A will be able to create useful internal representations by itself in an *early* stage of processing a

given sequence; it will often receive meaningful error signals long before errors of the first or second kind occur. These internal representations in turn must carry the discriminating information for enabling A to improve its low-level predictions. Therefore the chunker will receive fewer and fewer inputs, since more and more inputs become predictable by the automatizer. This is the *collapsing operation.* Ideally, the chunker will become obsolete after some time.

It must be emphasized that unlike with the incremental creation of a multi-level predictor hierarchy described in section 3, there is no formal proof that the 2-net *on-line* version is free of instabilities. One can imagine situations where A unlearns previously learned predictions because of the third term of its error function. Relative weighting of the different terms in A's error function represents an ad-hoc remedy for this potential problem. In the experiments below, relative weighting was not necessary.

## 5   EXPERIMENTS

One experiment with a multi-level chunking architecture involved a grammar which produced strings of many *a*'s and *b*'s such that there was local temporal structure within the training strings (see [6] for details). The task was to differentiate between strings with long overlapping suffixes. The conventional algorithm completely failed to solve the task; it became confused by the great numbers of input sequences with similar endings. Not so the chunking system: It soon discovered certain hierarchical temporal structures in the input sequences and decomposed the problem such that it was able to solve it within a few hundred-thousand training sequences.

The 2-net chunking system (the one with the potential for collapsing levels) was also tested against the conventional recurrent net algorithms. (See details in [26].) With the conventional algorithms, with various learning rates, and with more than 1,000,000 training sequences *performance did not improve in prediction tasks involving even as few as 20 time steps between relevant events.*

But, the 2-net chunking system was able to solve the task rather quickly. An efficient approximation of the BPTT-method was applied to both the chunker and the automatizer: *Only 3 iterations of error propagation 'back into the past' were performed at each time step.* Most of the test runs required less than 5000 training sequences. *Still the final weight matrix of the automatizer often resembled what one would hope to get from the conventional algorithm.* There were hidden units which learned to bridge the 20-step time lags by means of strong self-connections. The chunking system needed *less computation per time step than the conventional method and required many fewer training sequences.*

## 6   CONTINUOUS HISTORY COMPRESSION

The history compression technique formulated above defines expectation-mismatches in a yes-or-no fashion: Each input unit whose activation is not predictable at a certain time gives rise to an unexpected event. Each unexpected event provokes an update of the internal state of a higher-level predictor. The updates always take place according to the conventional activation spreading rules for re-

current neural nets. There is no concept of a partial mismatch or of a 'near-miss'. There is no possibility of updating the higher-level net 'just a little bit' in response to a 'nearly expected input'. In practical applications, some 'epsilon' has to be used to define an acceptable mismatch.

In reply to the above criticism, *continuous history compression* is based on the following ideas. In what follows, $v_i(t)$ denotes the $i$-th component of vector $v(t)$.

We use a local input representation. The components of $z^p(t)$ are forced to sum up to 1 and are interpreted as a prediction of the probability distribution of the possible $x^p(t+1)$. $z_j^p(t)$ is interpreted as the prediction of the probability that $x_j^p(t+1)$ is 1.

The output entropy

$$-\sum_j z_j^p(t) log\ z_j^p(t)$$

can be interpreted as a measure of the predictor's confidence. In the worst case, the predictor will expect every possible event with equal probability.

How much information (relative to the current predictor) is conveyed by the event $x_j^p(t+1) = 1$, once it is observed? According to [29] it is

$$-log\ z_j^p(t).$$

[28] defines update procedures based on Mozer's recent update function [12] that let highly informative events have a stronger influence on the history representation than less informative (more likely) events. The 'strength' of an update in response to a more or less unexpected event is a monotonically increasing function of the information the event conveys. One of the update procedures uses Pollack's recursive auto-associative memories [16] for storing unexpected events, thus yielding an entirely local learning algorithm for learning extended sequences.

## 7   ACKNOWLEDGEMENTS

Thanks to Josef Hochreiter for conducting the experiments. Thanks to Mike Mozer and Mark Ring for useful comments on an earlier draft of this paper. This research was supported in part by NSF PYI award IRI–9058450, grant 90–21 from the James S. McDonnell Foundation, and DEC external research grant 1250 to Michael C. Mozer.

## Footnotes

[1]See also [18] for a different hierarchical connectionist chunking system based on similar principles.

[2]Recently I became aware that Don Mathis had some related ideas (personal communication). A hierarchical approach to sequence *generation* was pursued by [9].

[3]For instance, we might employ the more limited feed-forward networks and a 'time window' approach. In this case, the number of previous inputs to be considered as a basis for the next prediction will remain fixed.

[4]A unique time representation is theoretically necessary to provide $P_{s+1}$ with unambiguous information about when the failure occurred (see also the last paragraph of section 2). A unique representation of the time that went by since the *last* unpredicted input occurred will do as well.

[5]In contrast, the reduced descriptions referred to by [11] are not unambiguous.

## References

[1] J. Bachrach. Learning to represent state, 1988. Unpublished master's thesis, University of Massachusetts, Amherst.

[2] U. Bodenhausen and A. Waibel. The tempo 2 algorithm: Adjusting time-delays by supervised learning. In D. S. Lippman, J. E. Moody, and D. S. Touretzky, editors, *Advances in Neural Information Processing Systems 3*, pages 155–161. San Mateo, CA: Morgan Kaufmann, 1991.

[3] J. L. Elman. Finding structure in time. Technical Report CRL Technical Report 8801, Center for Research in Language, University of California, San Diego, 1988.

[4] M. Gherrity. A learning algorithm for analog fully recurrent neural networks. In *IEEE/INNS International Joint Conference on Neural Networks, San Diego*, volume 1, pages 643–644, 1989.

[5] C. L. Giles and C. B. Miller. Learning and extracting finite state automata. Accepted for publication in *Neural Computation*, 1992.

[6] Josef Hochreiter. Diploma thesis, 1991. Institut für Informatik, Technische Universität München.

[7] M. I. Jordan. Serial order: A parallel distributed processing approach. Technical Report ICS Report 8604, Institute for Cognitive Science, University of California, San Diego, 1986.

[8] G. Lukes. Review of Schmidhuber's paper 'Recurrent networks adjusted by adaptive critics'. *Neural Network Reviews*, 4(1):41–42, 1990.

[9] Y. Miyata. An unsupervised PDP learning model for action planning. In *Proc. of the Tenth Annual Conference of the Cognitive Science Society, Hillsdale, NJ*, pages 223–229. Erlbaum, 1988.

[10] M. C. Mozer. A focused back-propagation algorithm for temporal sequence recognition. *Complex Systems*, 3:349–381, 1989.

[11] M. C. Mozer. Connectionist music composition based on melodic, stylistic, and psychophysical constraints. Technical Report CU-CS-495-90, University of Colorado at Boulder, 1990.

[12] M. C. Mozer. Induction of multiscale temporal structure. In D. S. Lippman, J. E. Moody, and D. S. Touretzky, editors, *Advances in Neural Information Processing Systems 4*, to appear. San Mateo, CA: Morgan Kaufmann, 1992.

[13] C. Myers. Learning with delayed reinforcement through attention-driven buffering. TR, Imperial College of Science, Technology and Medicine, 1990.

[14] B. A. Pearlmutter. Learning state space trajectories in recurrent neural networks. *Neural Computation*, 1:263–269, 1989.

[15] F. J. Pineda. Time dependent adaptive neural networks. In D. S. Touretzky, editor, *Advances in Neural Information Processing Systems 2*, pages 710–718. San Mateo, CA: Morgan Kaufmann, 1990.

[16] J. B. Pollack. Recursive distributed representation. *Artificial Intelligence*, 46:77–105, 1990.

[17] M. A. Ring. PhD Proposal: Autonomous construction of sensorimotor hierarchies in neural networks. Technical report, Univ. of Texas at Austin, 1990.

[18] M. A. Ring. Incremental development of complex behaviors through automatic construction of sensory-motor hierarchies. In L. Birnbaum and G. Collins, editors, *Machine Learning: Proceedings of the Eighth International Workshop*, pages 343–347. Morgan Kaufmann, 1991.

[19] A. J. Robinson and F. Fallside. The utility driven dynamic error propagation network. Technical Report CUED/F-INFENG/TR.1, Cambridge University Engineering Department, 1987.

[20] R. Rohwer. The 'moving targets' training method. In J. Kindermann and A. Linden, editors, *Proceedings of 'Distributed Adaptive Neural Information Processing', St.Augustin, 24.-25.5,*. Oldenbourg, 1989.

[21] D. E. Rumelhart, G. E. Hinton, and R. J. Williams. Learning internal representations by error propagation. In D. E. Rumelhart and J. L. McClelland, editors, *Parallel Distributed Processing*, volume 1, pages 318–362. MIT Press, 1986.

[22] J. H. Schmidhuber. A local learning algorithm for dynamic feedforward and recurrent networks. *Connection Science*, 1(4):403–412, 1989.

[23] J. H. Schmidhuber. Recurrent networks adjusted by adaptive critics. In *Proc. IEEE/INNS International Joint Conference on Neural Networks, Washington, D. C.*, volume 1, pages 719–722, 1990.

[24] J. H. Schmidhuber. Adaptive decomposition of time. In T. Kohonen, K. Mäkisara, O. Simula, and J. Kangas, editors, *Artificial Neural Networks*, pages 909–914. Elsevier Science Publishers B.V., North-Holland, 1991.

[25] J. H. Schmidhuber. A fixed size storage $O(n^3)$ time complexity learning algorithm for fully recurrent continually running networks. Accepted for publication in *Neural Computation*, 1992.

[26] J. H. Schmidhuber. Learning complex, extended sequences using the principle of history compression. Accepted for publication in *Neural Computation*, 1992.

[27] J. H. Schmidhuber. Learning to control fast-weight memories: An alternative to recurrent nets. Accepted for publication in *Neural Computation*, 1992.

[28] J. H. Schmidhuber, M. C. Mozer, and D. Prelinger. Continuous history compression. Technical report, Dept. of Comp. Sci., University of Colorado at Boulder, 1992.

[29] C. E. Shannon. A mathematical theory of communication (parts I and II). *Bell System Technical Journal*, XXVII:379–423, 1948.

[30] P. J. Werbos. Generalization of backpropagation with application to a recurrent gas market model. *Neural Networks*, 1, 1988.

[31] R. J. Williams. Toward a theory of reinforcement-learning connectionist systems. Technical Report NU-CCS-88-3, College of Comp. Sci., Northeastern University, Boston, MA, 1988.

[32] R. J. Williams. Complexity of exact gradient computation algorithms for recurrent neural networks. Technical Report Technical Report NU-CCS-89-27, Boston: Northeastern University, College of Computer Science, 1989.

[33] R. J. Williams and J. Peng. An efficient gradient-based algorithm for on-line training of recurrent network trajectories. *Neural Computation*, 4:491–501, 1990.

[34] R. J. Williams and D. Zipser. Experimental analysis of the real-time recurrent learning algorithm. *Connection Science*, 1(1):87–111, 1989.

[35] R. J. Williams and D. Zipser. Gradient-based learning algorithms for recurrent networks and their computational complexity. In *Back-propagation: Theory, Architectures and Applications*. Hillsdale, NJ: Erlbaum, 1992, in press.
